# Size Regularized Cut for Data Clustering

**Yixin Chen**
Department of CS
Univ. of New Orleans
yixin@cs.uno.edu

**Ya Zhang**
Department of EECS
Uinv. of Kansas
yazhang@ittc.ku.edu

**Xiang Ji**
NEC-Labs America, Inc.
xji@sv.nec-labs.com

## Abstract

We present a novel spectral clustering method that enables users to incorporate prior knowledge of the size of clusters into the clustering process. The cost function, which is named size regularized cut (SRcut), is defined as the sum of the inter-cluster similarity and a regularization term measuring the relative size of two clusters. Finding a partition of the data set to minimize SRcut is proved to be NP-complete. An approximation algorithm is proposed to solve a relaxed version of the optimization problem as an eigenvalue problem. Evaluations over different data sets demonstrate that the method is not sensitive to outliers and performs better than normalized cut.

## 1   Introduction

In recent years, spectral clustering based on graph partitioning theories has emerged as one of the most effective data clustering tools. These methods model the given data set as a weighted undirected graph. Each data instance is represented as a node. Each edge is assigned a weight describing the similarity between the two nodes connected by the edge. Clustering is then accomplished by finding the best cuts of the graph that optimize certain predefined cost functions. The optimization usually leads to the computation of the top eigenvectors of certain graph affinity matrices, and the clustering result can be derived from the obtained eigen-space [12, 6]. Many cost functions, such as the ratio cut [3], average association [15], spectral $k$-means [19], normalized cut [15], min-max cut [7], and a measure using conductance and cut [9] have been proposed along with the corresponding eigen-systems for the data clustering purpose.

The above data clustering methods, as well as most other methods in the literature, bear a common characteristic that manages to generate results maximizing the intra-cluster similarity, and/or minimizing the inter-cluster similarity. These approaches perform well in some cases, but fail drastically when target data sets possess complex, extreme data distributions, and when the user has special needs for the data clustering task. For example, it has been pointed out by several researchers that normalized cut sometimes displays sensitivity to outliers [7, 14]. Normalized cut tends to find a cluster consisting of a very small number of points if those points are far away from the center of the data set [14].

There has been an abundance of prior work on embedding user's prior knowledge of the data set in the clustering process. Kernighan and Lin [11] applied a local search procedure that maintained two equally sized clusters while trying to minimize the association between

the clusters. Wagstaff et al. [16] modified $k$-means method to deal with *a priori* knowledge about must-link and cannot link constraints. Banerjee and Ghosh [2] proposed a method to balance the size of the clusters by considering an explicit soft constraint. Xing et al. [17] presented a method to learn a clustering metric over user specified samples. Yu and Shi [18] introduced a method to include must-link grouping cues in normalized cut. Other related works include leaving $\epsilon$ fraction of the points unclustered to avoid the effect of outliers [4] and enforcing minimum cluster size constraint [10].

In this paper, we present a novel clustering method based on graph partitioning. The new method enables users to incorporate prior knowledge of the expected size of clusters into the clustering process. Specifically, the cost function of the new method is defined as the sum of the inter-cluster similarity and a regularization term that measures the relative size of two clusters. An "optimal" partition corresponds to a tradeoff between the inter-cluster similarity and the relative size of two clusters. We show that the size of the clusters generated by the optimal partition can be controlled by adjusting the weight on the regularization term. We also prove that the optimization problem is NP-complete. So we present an approximation algorithm and demonstrate its performance using two document data sets.

## 2   Size regularized cut

We model a given data set using a weighted undirected graph $\mathbb{G} = \mathbb{G}(\mathcal{V}, \mathcal{E}, \mathbf{W})$ where $\mathcal{V}$, $\mathcal{E}$, and $\mathbf{W}$ denote the vertex set, edge set, and graph affinity matrix, respectively. Each vertex $i \in \mathcal{V}$ represents a data point, and each edge $(i, j) \in \mathbb{E}$ is assigned a nonnegative weight $\mathbf{W}_{ij}$ to reflect the similarity between the data points $i$ and $j$. A graph partitioning method attempts to organize vertices into groups so that the intra-cluster similarity is high, and/or the inter-cluster similarity is low. A simple way to quantify the cost for partitioning vertices into two disjoint sets $\mathcal{V}_1$ and $\mathcal{V}_2$ is the cut size

$$\text{cut}(\mathcal{V}_1, \mathcal{V}_2) = \sum_{i \in \mathcal{V}_1, j \in \mathcal{V}_2} \mathbf{W}_{ij} \ ,$$

which can be viewed as the similarity or association between $\mathcal{V}_1$ and $\mathcal{V}_2$. Finding a binary partition of the graph that minimizes the cut size is known as the minimum cut problem. There exist efficient algorithms for solving this problem. However, the minimum cut criterion favors grouping small sets of isolated nodes in the graph [15].

To capture the need for more balanced clusters, it has been proposed to include the cluster size information as a multiplicative penalty factor in the cost function, such as average cut [3] and normalized cut [15]. Both cost functions can be uniformly written as [5]

$$\text{cost}(\mathcal{V}_1, \mathcal{V}_2) = \text{cut}(\mathcal{V}_1, \mathcal{V}_2) \left( \frac{1}{|\mathcal{V}_1|_{\boldsymbol{\beta}}} + \frac{1}{|\mathcal{V}_1|_{\boldsymbol{\beta}}} \right) \ . \tag{1}$$

Here, $\boldsymbol{\beta} = [\beta_1, \cdots, \beta_N]^T$ is a weight vector where $\beta_i$ is a nonnegative weight associated with vertex $i$, and $N$ is the total number of vertices in $\mathcal{V}$. The penalty factor for "unbalanced partition" is determined by $|\mathcal{V}_j|_{\boldsymbol{\beta}}$ $(j = 1, 2)$, which is a weighted cardinality (or weighted size) of $\mathcal{V}_j$, i.e.,

$$|\mathcal{V}_j|_{\boldsymbol{\beta}} = \sum_{i \in \mathcal{V}_j} \beta_i \ . \tag{2}$$

Dhillon [5] showed that if $\beta_i = 1$ (for all $i$), the cost function (1) becomes average cut. If $\beta_i = \sum_j \mathbf{W}_{ij}$, then (1) turns out to be normalized cut.

In contrast with minimum cut, average cut and normalized cut tend to generate more balanced clusters. However, due to the multiplicative nature of their cost functions, average cut and normalized cut are still sensitive to outliers. This is because the cut value for separating outliers from the rest of the data points is usually close to zero, and thus makes

the multiplicative penalty factor void. To avoid the drawback of the above multiplicative cost functions, we introduce an additive cost function for graph bi-partitioning. The cost function is named *size regularized cut* (SRcut), and is defined as

$$\mathrm{SRcut}(\mathcal{V}_1, \mathcal{V}_2) = \mathrm{cut}(\mathcal{V}_1, \mathcal{V}_2) - \alpha |\mathcal{V}_1|_{\boldsymbol{\beta}} |\mathcal{V}_2|_{\boldsymbol{\beta}} \tag{3}$$

where $|\mathcal{V}_j|_{\boldsymbol{\beta}}$ $(j = 1, 2)$ is described in (2), $\boldsymbol{\beta}$ and $\alpha > 0$ are given a priori. The last term in (3), $\alpha |\mathcal{V}_1|_{\boldsymbol{\beta}} |\mathcal{V}_2|_{\boldsymbol{\beta}}$, is the size regularization term, which can be interpreted as below.

Since $|\mathcal{V}_1|_{\boldsymbol{\beta}} + |\mathcal{V}_2|_{\boldsymbol{\beta}} = |\mathcal{V}|_{\boldsymbol{\beta}} = \boldsymbol{\beta}^T \mathbf{e}$ where $\mathbf{e}$ is a vector of 1's, it is straightforward to show that the following inequality $|\mathcal{V}_1|_{\boldsymbol{\beta}} |\mathcal{V}_2|_{\boldsymbol{\beta}} \leq \left( \frac{\boldsymbol{\beta}^T \mathbf{e}}{2} \right)^2$ holds for arbitrary $\mathcal{V}_1, \mathcal{V}_2 \in \mathcal{V}$ satisfying $\mathcal{V}_1 \cup \mathcal{V}_2 = \mathcal{V}$ and $\mathcal{V}_1 \cap \mathcal{V}_2 = \emptyset$. In addition, the equality holds if and only if

$$|\mathcal{V}_1|_{\boldsymbol{\beta}} = |\mathcal{V}_2|_{\boldsymbol{\beta}} = \frac{\boldsymbol{\beta}^T \mathbf{e}}{2}.$$

Therefore, $|\mathcal{V}_1|_{\boldsymbol{\beta}} |\mathcal{V}_2|_{\boldsymbol{\beta}}$ achieves the maximum value when two clusters are of equal weighted size. Consequently, *minimizing* SRcut *is equivalent to minimizing the similarity between two clusters and, at the same time, searching for a balanced partition*. The tradeoff between the inter-cluster similarity and the balance of the cut depends on the $\alpha$ parameter, which needs to be determined by the prior information on the size of clusters. If $\alpha = 0$, minimum SRcut will assign all vertices to one cluster. On the other end, if $\alpha \gg 0$, minimum SRcut will generate two clusters of equal size (if $N$ is an even number). We defer the discussion on the choice of $\alpha$ to Section 5.

In a spirit similar to that of (3), we can define *size regularized association* (SRassoc) as

$$\mathrm{SRassoc}(\mathcal{V}_1, \mathcal{V}_2) = \sum_{i=1,2} \mathrm{cut}(\mathcal{V}_i, \mathcal{V}_i) + 2\alpha |\mathcal{V}_1|_{\boldsymbol{\beta}} |\mathcal{V}_2|_{\boldsymbol{\beta}}$$

where $\mathrm{cut}(\mathcal{V}_i, \mathcal{V}_i)$ measures the intra-cluster similarity. An important property of SRassoc and SRcut is that they are naturally related:

$$\mathrm{SRcut}(\mathcal{V}_1, \mathcal{V}_2) = \frac{\mathrm{cut}(\mathcal{V}, \mathcal{V}) - \mathrm{SRassoc}(\mathcal{V}_1, \mathcal{V}_2)}{2} .$$

Hence, minimizing size regularized cut is in fact identical to maximizing size regularized association. In other words, *minimizing the size regularized inter-cluster similarity is equivalent to maximizing the size regularized intra-cluster similarity*. In this paper, we will use SRcut as the clustering criterion.

## 3   Size ratio monotonicity

Let $\mathcal{V}_1$ and $\mathcal{V}_2$ be a partition of $\mathcal{V}$. The size ratio $r = \frac{\min(|\mathcal{V}_1|_{\boldsymbol{\beta}}, |\mathcal{V}_2|_{\boldsymbol{\beta}})}{\max(|\mathcal{V}_1|_{\boldsymbol{\beta}}, |\mathcal{V}_2|_{\boldsymbol{\beta}})}$ defines the relative size of two clusters. It is always within the interval $[0, 1]$, and a larger value indicates a more balanced partition. The following theorem shows that by controlling the parameter $\alpha$ in the SRcut cost function, one can control the balance of the optimal partition. In addition, the size ratio increases monotonically as the increase of $\alpha$.

**Theorem 3.1 (Size Ratio Monotonicity)** *Let $\mathcal{V}_1^i$ and $\mathcal{V}_2^i$ be the clusters generated by the minimum* SRcut *with $\alpha = \alpha_i$, and the corresponding size ratio, $r_i$, be defined as*

$$r_i = \frac{\min(|\mathcal{V}_1^i|_{\boldsymbol{\beta}}, |\mathcal{V}_2^i|_{\boldsymbol{\beta}})}{\max(|\mathcal{V}_1^i|_{\boldsymbol{\beta}}, |\mathcal{V}_2^i|_{\boldsymbol{\beta}})} .$$

*If $\alpha_1 > \alpha_2 \geq 0$, then $r_1 \geq r_2$.*

**Proof:** Given vertex weight vector $\boldsymbol{\beta}$, let $\mathcal{S}$ be the collection of all distinct values that the size regularization term in (3) can have, i.e.,

$$\mathcal{S} = \{S \mid \mathcal{V}_1 \cup \mathcal{V}_2 = \mathcal{V}, \; \mathcal{V}_1 \cap \mathcal{V}_2 = \emptyset, \; S = |\mathcal{V}_1|_{\boldsymbol{\beta}}|\mathcal{V}_2|_{\boldsymbol{\beta}}\} \; .$$

Clearly, $|\mathcal{S}|$, the number of elements in $\mathcal{S}$, is less than or equal to $2^{N-1}$ where $N$ is the size of $\mathcal{V}$. Hence we can write the elements in $\mathcal{S}$ in ascending order as

$$0 = S_1 < S_2 < \cdots\cdots < S_{|\mathcal{S}|} \leq \left(\frac{\boldsymbol{\beta}^T \mathbf{e}}{2}\right)^2 \; .$$

Next, we define $\mathrm{cut}_i$ be the minimal cut satisfying $|\mathcal{V}_1|_{\boldsymbol{\beta}}|\mathcal{V}_2|_{\boldsymbol{\beta}} = S_i$, i.e.,

$$\mathrm{cut}_i = \min_{\substack{|\mathcal{V}_1|_{\boldsymbol{\beta}}|\mathcal{V}_2|_{\boldsymbol{\beta}} = S_i \\ \mathcal{V}_1 \cup \mathcal{V}_2 = \mathcal{V} \\ \mathcal{V}_1 \cap \mathcal{V}_2 = \emptyset}} \mathrm{cut}(\mathcal{V}_1, \mathcal{V}_2) \; ,$$

then

$$\min_{\substack{\mathcal{V}_1 \cup \mathcal{V}_2 = \mathcal{V} \\ \mathcal{V}_1 \cap \mathcal{V}_2 = \emptyset}} \mathrm{SRcut}(\mathcal{V}_1, \mathcal{V}_2) = \min_{i=1,\cdots,|\mathcal{S}|} (\mathrm{cut}_i - \alpha S_i) \; .$$

If $\mathcal{V}_1^2$ and $\mathcal{V}_2^2$ are the clusters generated by the minimum SRcut with $\alpha = \alpha_2$, then $|\mathcal{V}_1^2|_{\boldsymbol{\beta}}|\mathcal{V}_2^2|_{\boldsymbol{\beta}} = S_{k^*}$ where $k^* = \mathrm{argmin}_{i=1,\cdots,|\mathcal{S}|} (\mathrm{cut}_i - \alpha_2 S_i)$. Therefore, for any $1 \leq t < k^*$,

$$\mathrm{cut}_{k^*} - \alpha_2 S_{k^*} \leq \mathrm{cut}_t - \alpha_2 S_t \; . \tag{4}$$

If $\alpha_1 > \alpha_2$, we have

$$(\alpha_2 - \alpha_1) S_{k^*} < (\alpha_2 - \alpha_1) S_t \; . \tag{5}$$

Adding (4) and (5) gives $\mathrm{cut}_{k^*} - \alpha_1 S_{k^*} < \mathrm{cut}_t - \alpha_1 S_t$, which implies

$$k^* \leq \mathrm{argmin}_{i=1,\cdots,|\mathcal{S}|} (\mathrm{cut}_i - \alpha_1 S_i) \; . \tag{6}$$

Now, let $\mathcal{V}_1^1$ and $\mathcal{V}_2^1$ be the clusters generated by the minimum SRcut with $\alpha = \alpha_1$, and $|\mathcal{V}_1^1|_{\boldsymbol{\beta}}|\mathcal{V}_2^1|_{\boldsymbol{\beta}} = S_{j^*}$ where $j^* = \mathrm{argmin}_{i=1,\cdots,|\mathcal{S}|} (\mathrm{cut}_i - \alpha_1 S_i)$. From (6) we have $j^* \geq k^*$, therefore $S_{j^*} \geq S_{k^*}$, or equivalently $|\mathcal{V}_1^1|_{\boldsymbol{\beta}}|\mathcal{V}_2^1|_{\boldsymbol{\beta}} \geq |\mathcal{V}_1^2|_{\boldsymbol{\beta}}|\mathcal{V}_2^2|_{\boldsymbol{\beta}}$. Without loss of generality, we can assume that $|\mathcal{V}_1^1|_{\boldsymbol{\beta}} \leq |\mathcal{V}_2^1|_{\boldsymbol{\beta}}$ and $|\mathcal{V}_1^2|_{\boldsymbol{\beta}} \leq |\mathcal{V}_2^2|_{\boldsymbol{\beta}}$, therefore $|\mathcal{V}_1^1|_{\boldsymbol{\beta}} \leq \frac{|\mathcal{V}|_{\boldsymbol{\beta}}}{2}$ and $|\mathcal{V}_1^2|_{\boldsymbol{\beta}} \leq \frac{|\mathcal{V}|_{\boldsymbol{\beta}}}{2}$. Considering the fact that $f(x) = x(|\mathcal{V}|_{\boldsymbol{\beta}} - x)$ is strictly monotonically increasing as $x \leq \frac{|\mathcal{V}|_{\boldsymbol{\beta}}}{2}$ and $f(|\mathcal{V}_1^1|_{\boldsymbol{\beta}}) \geq f(|\mathcal{V}_1^2|_{\boldsymbol{\beta}})$, we have $|\mathcal{V}_1^1|_{\boldsymbol{\beta}} \geq |\mathcal{V}_1^2|_{\boldsymbol{\beta}}$. This leads to $r_1 = \frac{|\mathcal{V}_1^1|_{\boldsymbol{\beta}}}{|\mathcal{V}_2^1|_{\boldsymbol{\beta}}} \geq r_2 = \frac{|\mathcal{V}_1^2|_{\boldsymbol{\beta}}}{|\mathcal{V}_2^2|_{\boldsymbol{\beta}}}$ . $\qquad\square$

Unfortunately, minimizing size regularized cut for an arbitrary $\alpha$ is an NP-complete problem. This is proved in the following section.

## 4 Size regularized cut and graph bisection

The decision problem for minimum SRcut can be formulated as: whether, given an undirected graph $\mathbb{G}(\mathcal{V}, \mathcal{E}, \mathbf{W})$ with weight vector $\boldsymbol{\beta}$ and regularization parameter $\alpha$, a partition exists such that SRcut is less than a given cost. This decision problem is clearly NP because we can verify in polynomial time the SRcut value for a given partition. Next we show that graph bisection can be reduced, in polynomial time, to minimum SRcut. Since graph bisection is a classified NP-complete problem [1], so is minimum SRcut.

**Definition 4.1 (Graph Bisection)** *Given an undirected graph $\mathbb{G} = \mathbb{G}(\mathcal{V}, \mathcal{E}, \mathbf{W})$ with even number of vertices where $\mathbf{W}$ is the adjacency matrix, find a pair of disjoint subsets $\mathcal{V}_1, \mathcal{V}_2 \subset \mathcal{V}$ of equal size and $\mathcal{V}_1 \cup \mathcal{V}_2 = \mathcal{V}$, such that the number of edges between vertices in $\mathcal{V}_1$ and vertices in $\mathcal{V}_2$, i.e., $\mathrm{cut}(\mathcal{V}_1, \mathcal{V}_2)$, is minimal.*

**Theorem 4.2 (Reduction of Graph Bisection to SRcut)** *For any given undirected graph* $\mathbb{G} = \mathbb{G}(\mathcal{V}, \mathcal{E}, \mathbf{W})$ *where* $\mathbf{W}$ *is the adjacency matrix, finding the minimum bisection of* $\mathbb{G}$ *is equivalent to finding a partition of* $\mathbb{G}$ *that minimizes the SRcut cost function with weights* $\boldsymbol{\beta} = \mathbf{e}$ *and the regularization parameter* $\alpha > d^*$ *where*

$$d^* = \max_{i=1,\cdots,N} \sum_{j=1,\cdots,N} \mathbf{W}_{ij} .$$

**Proof:** Without loss of generality, we assume that $N$ is even (if not, we can always add an isolated vertex). Let $\text{cut}_i$ be the minimal cut with the size of the smaller subset is $i$, i.e.,

$$\text{cut}_i = \min_{\substack{\min(|\mathcal{V}_1|, |\mathcal{V}_2|) = i \\ \mathcal{V}_1 \cup \mathcal{V}_2 = \mathcal{V} \\ \mathcal{V}_1 \cap \mathcal{V}_2 = \emptyset}} \text{cut}(\mathcal{V}_1, \mathcal{V}_2) .$$

Clearly, we have $d^* \geq \text{cut}_{i+1} - \text{cut}_i$ for $0 \leq i \leq \frac{N}{2} - 1$. If $0 \leq i \leq \frac{N}{2} - 1$, then $N - 2i - 1 \geq 1$. Therefore, for any $\alpha > d^*$, we have

$$\alpha(N - 2i - 1) > d^* \geq \text{cut}_{i+1} - \text{cut}_i .$$

This implies that $\text{cut}_i - \alpha i(N - i) > \text{cut}_{i+1} - \alpha(i + 1)(N - i - 1)$ , or, equivalently,

$$\min_{\substack{\min(|\mathcal{V}_1|, |\mathcal{V}_2|) = i \\ \mathcal{V}_1 \cup \mathcal{V}_2 = \mathcal{V} \\ \mathcal{V}_1 \cap \mathcal{V}_2 = \emptyset}} \text{cut}(\mathcal{V}_1, \mathcal{V}_2) - \alpha |\mathcal{V}_1||\mathcal{V}_2| > \min_{\substack{\min(|\mathcal{V}_1|, |\mathcal{V}_2|) = i + 1 \\ \mathcal{V}_1 \cup \mathcal{V}_2 = \mathcal{V} \\ \mathcal{V}_1 \cap \mathcal{V}_2 = \emptyset}} \text{cut}(\mathcal{V}_1, \mathcal{V}_2) - \alpha |\mathcal{V}_1||\mathcal{V}_2|$$

for $0 \leq i \leq \frac{N}{2} - 1$. Hence, for any $\alpha > d^*$, minimizing SRcut is identical to minimizing

$$\text{cut}(\mathcal{V}_1, \mathcal{V}_2) - \alpha |\mathcal{V}_1||\mathcal{V}_2|$$

with the constraint that $|\mathcal{V}_1| = |\mathcal{V}_2| = \frac{N}{2}$, $\mathcal{V}_1 \cup \mathcal{V}_2 = \mathcal{V}$, and $\mathcal{V}_1 \cap \mathcal{V}_2 = \emptyset$, which is exactly the graph bisection problem since $\alpha |\mathcal{V}_1||\mathcal{V}_2| = \alpha \frac{N^2}{4}$ is a constant. $\qquad\square$

## 5  An approximation algorithm for SRcut

Given a partition of vertex set $\mathcal{V}$ into two sets $\mathcal{V}_1$ and $\mathcal{V}_2$, let $\mathbf{x} \in \{-1, 1\}^N$ be an indicator vector such that $x_i = 1$ if $i \in \mathcal{V}_1$ and $x_i = -1$ if $i \in \mathcal{V}_2$. It is not difficult to show that

$$\text{cut}(\mathcal{V}_1, \mathcal{V}_2) = \frac{(\mathbf{e} + \mathbf{x})^T}{2} \mathbf{W} \frac{(\mathbf{e} - \mathbf{x})}{2} \quad \text{and} \quad |\mathcal{V}_1|_{\boldsymbol{\beta}} |\mathcal{V}_2|_{\boldsymbol{\beta}} = \frac{(\mathbf{e} + \mathbf{x})^T}{2} \boldsymbol{\beta}\boldsymbol{\beta}^T \frac{(\mathbf{e} - \mathbf{x})}{2} .$$

We can therefore rewrite SRcut in (3) as a function of the indicator vector $\mathbf{x}$:

$$\text{SRcut}(\mathcal{V}_1, \mathcal{V}_2) = \frac{(\mathbf{e} + \mathbf{x})^T}{2} (\mathbf{W} - \alpha \boldsymbol{\beta}\boldsymbol{\beta}^T) \frac{(\mathbf{e} - \mathbf{x})}{2}$$

$$= -\frac{1}{4}\mathbf{x}^T(\mathbf{W} - \alpha \boldsymbol{\beta}\boldsymbol{\beta}^T)\mathbf{x} + \frac{1}{4}\mathbf{e}^T(\mathbf{W} - \alpha \boldsymbol{\beta}\boldsymbol{\beta}^T)\mathbf{e} . \tag{7}$$

Given $\mathbf{W}$, $\alpha$, and $\boldsymbol{\beta}$, we have

$$\text{argmin}_{\mathbf{x} \in \{-1,1\}^N} \text{SRcut}(\mathbf{x}) = \text{argmax}_{\mathbf{x} \in \{-1,1\}^N} \mathbf{x}^T(\mathbf{W} - \alpha \boldsymbol{\beta}\boldsymbol{\beta}^T)\mathbf{x}$$

If we define a normalized indicator vector, $\mathbf{y} = \frac{1}{\sqrt{N}}\mathbf{x}$ (i.e., $\|\mathbf{y}\| = 1$), then minimum SRcut can be found by solving the following discrete optimization problem

$$\mathbf{y} = \text{argmax}_{\mathbf{y} \in \{-\frac{1}{\sqrt{N}}, \frac{1}{\sqrt{N}}\}^N} \mathbf{y}^T(\mathbf{W} - \alpha \boldsymbol{\beta}\boldsymbol{\beta}^T)\mathbf{y} , \tag{8}$$

which is NP-complete. However, if we relax all the elements in the indicator vector $\mathbf{y}$ from discrete values to real values and keep the unit length constraint on $\mathbf{y}$, the above optimization problem can be easily solved. And the solution is the eigenvector corresponding to the largest eigenvalue of $\mathbf{W} - \alpha \boldsymbol{\beta}\boldsymbol{\beta}^T$ (or named the largest eigenvector).

Similar to other spectral graph partitioning techniques that use top eigenvectors to approximate "optimal" partitions, the largest eigenvector of $\mathbf{W} - \alpha\boldsymbol{\beta}\boldsymbol{\beta}^T$ provides a linear search direction, along which a splitting point can be found. We use a simple approach by checking each element in the largest eigenvector as a possible splitting point. The vertices, whose continuous indicators are greater than or equal to the splitting point, are assigned to one cluster. The remaining vertices are assigned to the other cluster. The corresponding SRcut value is then computed. The final partition is determined by the splitting point with the minimum SRcut value. The relaxed optimization problem provides a lower bound on the optimal SRcut value, SRcut*. Let $\lambda_1$ be the largest eigenvalue of $\mathbf{W} - \alpha\boldsymbol{\beta}\boldsymbol{\beta}^T$. From (7) and (8), it is straightforward to show that

$$\text{SRcut}^* \geq \frac{\mathbf{e}^T(\mathbf{W} - \alpha\boldsymbol{\beta}\boldsymbol{\beta}^T)\mathbf{e} - N\lambda_1}{4} \ .$$

The SRcut value of the partition generated by the largest eigenvector provides an upper bound for SRcut*.

As implied by SRcut cost function in (3), the partition of the dataset depends on the value of $\alpha$, which determines the tradeoff between inter-cluster similarity and the balance of the partition. Moreover, Theorem 3.1 indicates that with the increase of $\alpha$, the size ratio of the clusters generated by the optimal partition increase monotonically, i.e., the partition becomes more balanced. Even though, we do not have a counterpart of Theorem 3.1 for the approximated partition derived above, our empirical study shows that, in general, the size ratio of the approximated partition increases along with $\alpha$. Therefore, we use the prior information on the size of the clusters to select $\alpha$. Specifically, we define expected size ratio, $R$, as $R = \frac{\min(s_1, s_2)}{\max(s_1, s_2)}$ where $s_1$ and $s_2$ are the expected size of the two clusters (known a priori). We then search for a value of $\alpha$ such that the resulting size ratio is close to $R$. A simple one-dimensional search method based on bracketing and bisection is implemented [13]. The pseudo code of the searching algorithm is given in Algorithm 1 along with the rest of the clustering procedure. The input of the algorithm is the graph affinity matrix $\mathbf{W}$, the weight vector $\boldsymbol{\beta}$, the expected size ratio $R$, and $\alpha_0 > 0$ (the initial value of $\alpha$). The output is a partition of $\mathcal{V}$. In our experiments, $\alpha_0$ is chosen to be $10\frac{\mathbf{e}^T\mathbf{W}\mathbf{e}}{N^2}$.

If the expected size ratio $R$ is unknown, one can estimate $R$ assuming that the data are i.i.d. samples and a sample belongs to the smaller cluster with probability $p \leq 0.5$ (i.e., $R = \frac{p}{1-p}$). It is not difficult to prove that $\hat{p}$ of $n$ randomly selected samples from the data set is an unbiased estimator of $p$. Moreover, the distribution of $\hat{p}$ can be well approximated by a normal distribution with mean $p$ and variance $\frac{p(1-p)}{n}$ when $n$ is sufficiently large (say $n > 30$). Hence $\hat{p}$ converges to $p$ as the increase of $n$. This suggests a simple strategy for SRcut with unknown $R$. One can manually examine $n \ll N$ randomly selected data instances to get $\hat{p}$ and the 95% confidence interval $[p_{low}, p_{high}]$, from which one can evaluate the invertal $[R_{low}, R_{high}]$ for $R$. Algorithm 1 is then applied to a number of evenly distributed $R$'s within the interval to find the corresponding partitions. The final partition is chosen to be the one with the minimum cut value by assuming that a "good" partition should have a small cut.

# 6 Time complexity

The time complexity of each iteration is determined by that of computing the largest eigenvector. Using power method or Lanczos method [8], the running time is $O(MN^2)$ where $M$ is the number of matrix-vector computations required and $N$ is the number of vertices. Hence the overall time complexity is $O(KMN^2)$ where $K$ is the number of iterations in searching $\alpha$. Similar to other spectral graph clustering methods, the time complexity of SRcut can be significantly reduced if the affinity matrix $\mathbf{W}$ is sparse, i.e., the graph is only

**Algorithm 1: Size Regularized Cut**

```
 1 initialize α_l to 2α_0 and α_h to α_0/2
 2 REPEAT
 3    α_l ← α_l/2, y ← largest eigenvector of W − α_l ββ^T
 4    partition V using y and compute size ratio r
 5 UNTIL (r < R)
 6 REPEAT
 7    α_h ← 2α_h, y ← largest eigenvector of W − α_h ββ^T
 8    partition V using y and compute size ratio r
 9 UNTIL (r ≥ R)
10 REPEAT
11    α ← (α_l+α_h)/2, y ← largest eigenvector of W − αββ^T
12    partition V using y and compute size ratio r
13    IF (r < R)
14       α_l ← α
15    ELSE
16       α_h ← α
17    END IF
18 UNTIL (|r − R| < 0.01R or α_h − α_l < 0.01α_0)
```

locally connected. Although $\mathbf{W} - \alpha\boldsymbol{\beta}\boldsymbol{\beta}^T$ is in general not sparse, the time complexity of power method is still $O(MN)$. This is because $(\mathbf{W} - \alpha\boldsymbol{\beta}\boldsymbol{\beta}^T)\mathbf{y}$ can be evaluated as the sum of $\mathbf{W}\mathbf{y}$ and $\alpha\boldsymbol{\beta}(\boldsymbol{\beta}^T\mathbf{y})$, each requiring $O(N)$ operations. Therefore, by enforcing the sparsity, the overall time complexity of SRcut is $O(KMN)$.

# 7 Experiments

We test the SRcut algorithm using two data sets, Reuters-21578 document corpus and 20-Newsgroups. Reuters-21578 data set contains 21578 documents that have been manually assigned to 135 topics. In our experiments, we discarded documents with multiple category labels, and removed the topic classes containing less than 5 documents. This leads to a data set of 50 clusters with a total of 9102 documents. The 20-Newsgroups data set contains about 20000 documents collected from 20 newsgroups, each corresponding to a distinct topic. The number of news articles in each cluster is roughly the same. We pair each cluster with another cluster to form a data set, so that 190 test data sets are generated. Each document is represented by a term-frequency vector using TF-IDF weights.

We use the normalized mutual information as our evaluation metric. Normalized mutual information is always within the interval $[0, 1]$, with a larger value indicating a better performance. A simple sampling scheme described in Section 5 is used to estimate the expected size ratio. For the Reuters-21578 data set, 50 test runs were conducted, each on a test set created by mixing 2 topics randomly selected from the data set. The performance score in Table 1 was obtained by averaging the scores from 50 test runs. The results for 20-Newsgroups data set were obtained by averaging the scores from 190 test data sets. Clearly, SRcut outperforms the normalized cut on both data sets. SRcut performs significantly better than normalized cut on the 20-Newsgroups data set. In comparison with Reuters-21578, many topic classes in the 20-Newsgroups data set contain outliers. The results suggest that SRcut is less sensitive to outliers than normalized cut.

# 8 Conclusions

We proposed size regularized cut, a novel method that enables users to specify prior knowledge of the size of two clusters in spectral clustering. The SRcut cost function takes into

Table 1: Performance comparison for SRcut and Normalized Cut. The numbers shown are the normalized mutual information. A larger value indicates a better performance.

| Algorithms | Reuters-21578 | 20-Newsgroups |
|---|---|---|
| **SRcut** | **0.7330** | **0.7315** |
| Normalized Cut | 0.7102 | 0.2531 |

account inter-cluster similarity and the relative size of two clusters. The "optimal" partition of the data set corresponds to a tradeoff between the inter-cluster similarity and the balance of the partition. We proved that finding a partition with minimum SRcut is an NP-complete problem. We presented an approximation algorithm to solve a relaxed version of the optimization problem. Evaluations over different data sets indicate that the method is not sensitive to outliers and performs better than normalized cut. The SRcut model can be easily adapted to solve multiple-clusters problem by applying the clustering method recursively/iteratively on data sets. Since graph bisection can be reduced to SRcut, the proposed approximation algorithm provides a new spectral technique for graph bisection. Comparing SRcut with other graph bisection algorithms is therefore an interesting future work.

## References

[1] S. Arora, D. Karger, and M. Karpinski, "Polynomial Time Approximation Schemes for Dense Instances of NP-hard Problems," *Proc. ACM Symp. on Theory of Computing*, pp. 284-293, 1995.

[2] A. Banerjee and J. Ghosh, "On Scaling up Balanced Clustering Algorithms," *Proc. SIAM Int'l Conf. on Data Mining*, pp. 333-349, 2002.

[3] P. K. Chan, D. F. Schlag, and J. Y. Zien, "Spectral k-Way Ratio-Cut Partitioning and Clustering," *IEEE Trans. on Computer-Aided Design of Integrated Circuits and Systems*, 13:1088-1096, 1994.

[4] M. Charikar, S. Khuller, D. M. Mount, and G. Narasimhan, "Algorithms for Facility Location Problems with Outliers," *Proc. ACM-SIAM Symp. on Discrete Algorithms*, pp. 642-651, 2001.

[5] I. S. Dhillon, "Co-clustering Documents and Words using Bipartite Spectral Graph Partitioning," *Proc. ACM SIGKDD Conf. Knowledge Discovery and Data Mining*, pp. 269-274, 2001.

[6] C. Ding, "Data Clustering: Principal Components, Hopfield and Self-Aggregation Networks," *Proc. Int'l Joint Conf. on Artificial Intelligence*, pp. 479-484, 2003.

[7] C. Ding, X. He, H. Zha, M. Gu, and H. Simon, "Spectral Min-Max Cut for Graph Partitioning and Data Clustering," *Proc. IEEE Int'l Conf. Data Mining*, pp. 107-114, 2001.

[8] G. H. Golub and C. F. Van Loan, *Matrix Computations*, John Hopkins Press, 1999.

[9] R. Kannan, S. Vempala, and A. Vetta, "On Clusterings - Good, Bad and Spectral," *Proc. IEEE Symp. on Foundations of Computer Science*, pp. 367-377, 2000.

[10] D. R. Karget and M. Minkoff, "Building Steiner Trees with Incomplete Global Knowledge," *Proc. IEEE Symp. on Foundations of Computer Science*, pp. 613-623, 2000

[11] B. Kernighan and S. Lin, "An Efficient Heuristic Procedure for Partitioning Graphs," *The Bell System Technical Journal*, 49:291-307, 1970.

[12] A. Y. Ng, M. I. Jordan, and Y. Weiss, "On Spectral Clustering: Analysis and an Algorithm," *Advances in Neural Information Processing Systems 14*, pp. 849-856, 2001.

[13] W. H. Press, S. A. Teukolsky, W. T. Vetterling, and B. P. Flannery, *Numerical Recipes in C*, second edition, Cambridge University Press, 1992.

[14] A. Rahimi and B. Recht, "Clustering with Normalized Cuts is Clustering with a Hyperplane," *Statistical Learning in Computer Vision*, 2004.

[15] J. Shi and J. Malik, "Normalized Cuts and Image Segmentation," *IEEE Trans. on Pattern Analysis and Machine Intelligence*, 22:888-905, 2000.

[16] K. Wagstaff, C. Cardie, S. Rogers, and S. Schrodl, "Constrained K-means Clustering with Background Knowledge," *Proc. Int'l Conf. on Machine Learning*, pp. 577-584, 2001.

[17] E. P. Xing, A. Y. Ng, M. I. Jordan, and S. Russell, "Distance Metric Learning, with Applications to Clustering with Side Information," *Advances in Neural Information Processing Systems 15*, pp. 505-512, 2003.

[18] X. Yu and J. Shi, "Segmentation Given Partial Grouping Constraints," *IEEE Trans. on Pattern Analysis and Machine Intelligence*, 26:173-183, 2004.

[19] H. Zha, X. He, C. Ding, H. Simon, and M. Gu, "Spectral Relaxation for K-means Clustering," *Advances in Neural Information Processing Systems 14*, pp. 1057-1064, 2001.
